# Implicit Wiener Series for Higher-Order Image Analysis

**Matthias O. Franz**     **Bernhard Schölkopf**
Max-Planck-Institut für biologische Kybernetik
Spemannstr. 38, D-72076 Tübingen, Germany
mof;bs@tuebingen.mpg.de

## Abstract

The computation of classical higher-order statistics such as higher-order moments or spectra is difficult for images due to the huge number of terms to be estimated and interpreted. We propose an alternative approach in which multiplicative pixel interactions are described by a series of Wiener functionals. Since the functionals are estimated implicitly via polynomial kernels, the combinatorial explosion associated with the classical higher-order statistics is avoided. First results show that image structures such as lines or corners can be predicted correctly, and that pixel interactions up to the order of five play an important role in natural images.

Most of the interesting structure in a natural image is characterized by its higher-order statistics. Arbitrarily oriented lines and edges, for instance, cannot be described by the usual pairwise statistics such as the power spectrum or the autocorrelation function: From knowing the intensity of one point on a line alone, we cannot predict its neighbouring intensities. This would require knowledge of a second point on the line, i.e., we have to consider some third-order statistics which describe the interactions between triplets of points. Analogously, the prediction of a corner neighbourhood needs at least fourth-order statistics, and so on.

In terms of Fourier analysis, higher-order image structures such as edges or corners are described by *phase alignments*, i.e. phase correlations between several Fourier components of the image. Classically, harmonic phase interactions are measured by higher-order spectra [4]. Unfortunately, the estimation of these spectra for high-dimensional signals such as images involves the estimation and interpretation of a huge number of terms. For instance, a sixth-order spectrum of a $16\times16$ sized image contains roughly $10^{12}$ coefficients, about $10^{10}$ of which would have to be estimated independently if all symmetries in the spectrum are considered. First attempts at estimating the higher-order structure of natural images were therefore restricted to global measures such as skewness or kurtosis [8], or to submanifolds of fourth-order spectra [9].

Here, we propose an alternative approach that models the interactions of image points in a series of *Wiener functionals*. A Wiener functional of order $n$ captures those image components that can be predicted from the multiplicative interaction of $n$ image points. In contrast to higher-order spectra or moments, the estimation of a Wiener model does not require the estimation of an excessive number of terms since it can be computed implicitly

via polynomial kernels. This allows us to decompose an image into components that are characterized by interactions of a given order.

In the next section, we introduce the Wiener expansion and discuss its capability of modeling higher-order pixel interactions. The implicit estimation method is described in Sect. 2, followed by some examples of use in Sect. 3. We conclude in Sect. 4 by briefly discussing the results and possible improvements.

## 1  Modeling pixel interactions with Wiener functionals

For our analysis, we adopt a prediction framework: Given a $d \times d$ neighbourhood of an image pixel, we want to predict its gray value from the gray values of the neighbours. We are particularly interested to which extent interactions of different orders contribute to the overall prediction. Our basic assumption is that the dependency of the central pixel value $y$ on its neighbours $x_i$, $i = 1, \ldots, m = d^2 - 1$ can be modeled as a series

$$y = H_0[\mathbf{x}] + H_1[\mathbf{x}] + H_2[\mathbf{x}] + \cdots + H_n[\mathbf{x}] + \cdots \tag{1}$$

of discrete *Volterra functionals*

$$H_0[\mathbf{x}] = h_0 = \text{const.} \quad \text{and} \quad H_n[\mathbf{x}] = \sum_{i_1=1}^{m} \cdots \sum_{i_n=1}^{m} h_{i_1 \ldots i_n}^{(n)} x_{i_1} \ldots x_{i_n}. \tag{2}$$

Here, we have stacked the grayvalues of the neighbourhood into the vector $\mathbf{x} = (x_1, \ldots, x_m)^{\top} \in \mathbb{R}^m$. The discrete $n$th-order Volterra functional is, accordingly, a linear combination of all ordered $n$th-order monomials of the elements of $\mathbf{x}$ with $m^n$ coefficients $h_{i_1 \ldots i_n}^{(n)}$. Volterra functionals provide a controlled way of introducing multiplicative interactions of image points since a functional of order $n$ contains all products of the input of order $n$. In terms of higher-order statistics, this means that we can control the order of the statistics used since an $n$th-order Volterra series leads to dependencies between maximally $n + 1$ pixels.

Unfortunately, Volterra functionals are not orthogonal to each other, i.e., depending on the input distribution, a functional of order $n$ generally leads to additional lower-order interactions. As a result, the output of the functional will contain components that are proportional to that of some lower-order monomials. For instance, the output of a second-order Volterra functional for Gaussian input generally has a mean different from zero [5]. If one wants to estimate the zeroeth-order component of an image (i.e., the constant component created without pixel interactions) the constant component created by the second-order interactions needs to be subtracted. For general Volterra series, this correction can be achieved by decomposing it into a new series $y = G_0[\mathbf{x}] + G_1[\mathbf{x}] + \cdots + G_n[\mathbf{x}] + \cdots$ of functionals $G_n[\mathbf{x}]$ that are uncorrelated, i.e., orthogonal with respect to the input. The resulting *Wiener functionals*[1] $G_n[\mathbf{x}]$ are linear combinations of Volterra functionals up to order $n$. They are computed from the original Volterra series by a procedure akin to Gram-Schmidt orthogonalization [5]. It can be shown that any Wiener expansion of finite degree minimizes the mean squared error between the true system output and its Volterra series model [5]. The orthogonality condition ensures that a Wiener functional of order $n$ captures only the component of the image created by the multiplicative interaction of $n$ pixels. In contrast to general Volterra functionals, a Wiener functional is orthogonal to all monomials of lower order [5].

So far, we have not gained anything compared to classical estimation of higher-order moments or spectra: an $n$th-order Volterra functional contains the same number of terms as

the corresponding $n + 1$-order spectrum, and a Wiener functional of the same order has an even higher number of coefficients as it consists also of lower-order Volterra functionals. In the next section, we will introduce an implicit representation of the Wiener series using polynomial kernels which allows for an efficient computation of the Wiener functionals.

## 2 Estimating Wiener series by regression in RKHS

**Volterra series as linear functionals in RKHS.**  The $n$th-order Volterra functional is a weighted sum of all $n$th-order monomials of the input vector $\mathbf{x}$. We can interpret the evaluation of this functional for a given input $\mathbf{x}$ as a map $\phi_n$ defined for $n = 0, 1, 2, \ldots$ as

$$\phi_0(\mathbf{x}) = 1 \quad \text{and} \quad \phi_n(\mathbf{x}) = (x_1^n, x_1^{n-1}x_2, \ldots, x_1 x_2^{n-1}, x_2^n, \ldots, x_m^n) \tag{3}$$

such that $\phi_n$ maps the input $\mathbf{x} \in \mathbb{R}^m$ into a vector $\phi_n(\mathbf{x}) \in \mathbb{F}_n = \mathbb{R}^{m^n}$ containing all $m^n$ ordered monomials of degree $n$. Using $\phi_n$, we can write the $n$th-order Volterra functional in Eq. (2) as a scalar product in $\mathbb{F}_n$,

$$H_n[\mathbf{x}] = \eta_n^\top \phi_n(\mathbf{x}), \tag{4}$$

with the coefficients stacked into the vector $\eta_n = (h_{1,1,..1}^{(n)}, h_{1,2,..1}^{(n)}, h_{1,3,..1}^{(n)}, \ldots)^\top \in \mathbb{F}_n$. The same idea can be applied to the entire $p$th-order Volterra series. By stacking the maps $\phi_n$ into a single map $\phi^{(p)}(\mathbf{x}) = (\phi_0(\mathbf{x}), \phi_1(\mathbf{x}), \ldots, \phi_p(\mathbf{x}))^\top$, one obtains a mapping from $\mathbb{R}^m$ into $\mathbb{F}^{(p)} = \mathbb{R} \times \mathbb{R}^m \times \mathbb{R}^{m^2} \times \ldots \mathbb{R}^{m^p} = \mathbb{R}^M$ with dimensionality $M = \frac{1-m^{p+1}}{1-m}$. The entire $p$th-order Volterra series can be written as a scalar product in $\mathbb{F}^{(p)}$

$$\sum\nolimits_{n=0}^{p} H_n[\mathbf{x}] = (\eta^{(p)})^\top \phi^{(p)}(\mathbf{x}) \tag{5}$$

with $\eta^{(p)} \in \mathbb{F}^{(p)}$. Below, we will show how we can express $\eta^{(p)}$ as an expansion in terms of the training points. This will dramatically reduce the number of parameters we have to estimate.

This procedure works because the space $\mathbb{F}_n$ of $n$th-order monomials has a very special property: it has the structure of a *reproducing kernel Hilbert space (RKHS)*. As a consequence, the dot product in $\mathbb{F}_n$ can be computed by evaluating a positive definite kernel function $k_n(\mathbf{x}_1, \mathbf{x}_2)$. For monomials, one can easily show that (e.g., [6])

$$\phi_n(\mathbf{x}_1)^\top \phi_n(\mathbf{x}_2) = (\mathbf{x}_1^\top \mathbf{x}_2)^n =: k_n(\mathbf{x}_1, \mathbf{x}_2). \tag{6}$$

Since $\mathbb{F}^{(p)}$ is generated as a direct sum of the single spaces $\mathbb{F}_n$, the associated scalar product is simply the sum of the scalar products in the $\mathbb{F}_n$:

$$\phi^{(p)}(\mathbf{x}_1)^\top \phi^{(p)}(\mathbf{x}_2) = \sum\nolimits_{n=0}^{p} (\mathbf{x}_1^\top \mathbf{x}_2)^n = k^{(p)}(\mathbf{x}_1, \mathbf{x}_2). \tag{7}$$

Thus, we have shown that the discretized Volterra series can be expressed as a linear functional in a RKHS[2].

**Linear regression in RKHS.**  For our prediction problem (1), the RKHS property of the Volterra series leads to an efficient solution which is in part due to the so called *representer theorem* (e.g., [6]). It states the following: suppose we are given $N$ observations

$(\mathbf{x}_1, y_1), \ldots, (\mathbf{x}_N, y_N)$ of the function (1) and an arbitrary cost function $c$, $\Omega$ is a nonde-creasing function on $\mathbb{R}_{>0}$ and $\|.\|_{\mathbb{F}}$ is the norm of the RKHS associated with the kernel $k$. If we minimize an objective function

$$c((\mathbf{x}_1, y_1, f(\mathbf{x}_1)), \ldots, (\mathbf{x}_N, y_N, f(\mathbf{x}_N))) + \Omega(\|f\|_{\mathbb{F}}), \qquad (8)$$

over all functions in the RKHS, then an optimal solution[3] can be expressed as

$$f(\mathbf{x}) = \sum_{j=1}^{N} a_j k(\mathbf{x}, \mathbf{x}_j), \quad a_j \in \mathbb{R}. \qquad (9)$$

In other words, although we optimized over the entire RKHS including functions which are defined for arbitrary input points, it turns out that we can always express the solution in terms of the observations $\mathbf{x}_j$ only. Hence the optimization problem over the extremely large number of coefficients $\eta^{(p)}$ in Eq. (5) is transformed into one over $N$ variables $a_j$.

Let us consider the special case where the cost function is the mean squared error, $c((\mathbf{x}_1, y_1, f(\mathbf{x}_1)), \ldots, (\mathbf{x}_N, y_N, f(\mathbf{x}_N))) = \frac{1}{N} \sum_{j=1}^{N}(f(\mathbf{x}_j) - y_j)^2$, and the regularizer $\Omega$ is zero[4]. The solution for $\mathbf{a} = (a_1, \ldots, a_N)$ is readily computed by setting the derivative of (8) with respect to the vector $\mathbf{a}$ equal to zero; it takes the form $\mathbf{a} = K^{-1}\mathbf{y}$ with the *Gram matrix* defined as $K_{ij} = k(\mathbf{x}_i, \mathbf{x}_j)$, hence[5]

$$y = f(\mathbf{x}) = \mathbf{a}^{\top}\mathbf{z}(\mathbf{x}) = \mathbf{y}^{\top}K^{-1}\mathbf{z}(\mathbf{x}), \qquad (10)$$

where $\mathbf{z}(\mathbf{x}) = (k(\mathbf{x}, \mathbf{x}_1), k(\mathbf{x}, \mathbf{x}_2), \ldots k(\mathbf{x}, \mathbf{x}_N))^{\top} \in \mathbb{R}^N$.

**Implicit Wiener series estimation.** As we stated above, the $p$th-degree Wiener expansion is the $p$th-order Volterra series that minimizes the squared error. This can be put into the regression framework: since any finite Volterra series can be represented as a linear functional in the corresponding RKHS, we can find the $p$th-order Volterra series that minimizes the squared error by linear regression. This, by definition, must be the $p$th-degree Wiener series since no other Volterra series has this property[6]. From Eqn. (10), we obtain the following expressions for the implicit Wiener series

$$G_0[\mathbf{x}] = \frac{1}{N}\mathbf{y}^{\top}\mathbf{1}, \quad \sum_{n=0}^{p} G_n[\mathbf{x}] = \sum_{n=0}^{p} H_n[\mathbf{x}] = \mathbf{y}^{\top}K_p^{-1}\mathbf{z}^{(p)}(\mathbf{x}) \qquad (11)$$

where the Gram matrix $K_p$ and the coefficient vector $\mathbf{z}^{(p)}(\mathbf{x})$ are computed using the kernel from Eq. (7) and $\mathbf{1} = (1, 1, \ldots)^{\top} \in \mathbb{R}^N$. Note that the Wiener series is represented only implicitly since we are using the RKHS representation as a sum of scalar products with the training points. Thus, we can avoid the "curse of dimensionality", i.e., there is no need to compute the possibly large number of coefficients explicitly.

The explicit Volterra and Wiener expansions can be recovered from Eq. (11) by collecting all terms containing monomials of the desired order and summing them up. The individual $n$th-order Volterra functionals in a Wiener series of degree $p > 0$ are given implicitly by

$$H_n[\mathbf{x}] = \mathbf{y}^{\top}K_p^{-1}\mathbf{z}_n(\mathbf{x}) \qquad (12)$$

with $\mathbf{z}_n(\mathbf{x}) = ((\mathbf{x}_1^{\top}\mathbf{x})^n, (\mathbf{x}_2^{\top}\mathbf{x})^n, \ldots, (\mathbf{x}_N^{\top}\mathbf{x})^n)^{\top}$. For $p = 0$ the only term is the constant zero-order Volterra functional $H_0[\mathbf{x}] = G_0[\mathbf{x}]$. The coefficient vector $\eta_n = (h_{1,1,\ldots 1}^{(n)}, h_{1,2,\ldots 1}^{(n)}, h_{1,3,\ldots 1}^{(n)}, \ldots)^{\top}$ of the explicit Volterra functional is obtained as

$$\eta_n = \Phi_n^{\top}K_p^{-1}\mathbf{y} \qquad (13)$$

using the design matrix $\Phi_n = (\phi_n(\mathbf{x}_1)^\top, \phi_n(\mathbf{x}_1)^\top, \ldots, \phi_n(\mathbf{x}_1)^\top)^\top$. The individual Wiener functionals can only be recovered by applying the regression procedure twice. If we are interested in the $n$th-degree Wiener functional, we have to compute the solution for the kernels $k^{(n)}(\mathbf{x}_1, \mathbf{x}_2)$ and $k^{(n-1)}(\mathbf{x}_1, \mathbf{x}_2)$. The Wiener functional for $n > 0$ is then obtained from the difference of the two results as

$$G_n[\mathbf{x}] = \sum\nolimits_{i=0}^{n} G_i[\mathbf{x}] - \sum\nolimits_{i=0}^{n-1} G_i[\mathbf{x}] = \mathbf{y}^\top \left[ K_n^{-1}\, \mathbf{z}^{(n)}(\mathbf{x}) - K_{n-1}^{-1}\, \mathbf{z}^{(n-1)}(\mathbf{x}) \right]. \quad (14)$$

The corresponding $i$th-order Volterra functionals of the $n$th-degree Wiener functional are computed analogously to Eqns. (12) and (13) [3].

**Orthogonality.** The resulting Wiener functionals must fulfill the orthogonality condition which in its strictest form states that a $p$th-degree Wiener functional must be orthogonal to all monomials in the input of lower order. Formally, we will prove the following

**Theorem 1** *The functionals obtained from Eq. (14) fulfill the orthogonality condition*

$$E\left[m(\mathbf{x})G_p[\mathbf{x}]\right] = 0 \quad (15)$$

*where $E$ denotes the expectation over the input distribution and $m(\mathbf{x})$ an arbitrary $i$th-order monomial with $i < p$.*

We will show that this a consequence of the least squares fit of any linear expansion in a set of basis functions of the form $y = \sum_{j=1}^{M} \gamma_j \varphi_j(\mathbf{x})$. In the case of the Wiener and Volterra expansions, the basis functions $\varphi_j(\mathbf{x})$ are monomials of the components of $\mathbf{x}$.

We denote the error of the expansion as $e(\mathbf{x}) = y - \sum_{j=1}^{M} \gamma_j \varphi_j(\mathbf{x}_i)$. The minimum of the expected quadratic loss $L$ with respect to the expansion coefficient $\gamma_k$ is given by

$$\frac{\partial L}{\partial \gamma_k} = \frac{\partial}{\partial \gamma_k} E\|e(\mathbf{x})\|^2 = -2E\left[\varphi_k(\mathbf{x})e(\mathbf{x})\right] = 0. \quad (16)$$

This means that, for an expansion in a set of basis functions minimizing the squared error, the error is orthogonal to all basis functions used in the expansion.

Now let us assume we know the Wiener series expansion (which minimizes the mean squared error) of a system up to degree $p - 1$. The approximation error is given by the sum of the higher-order Wiener functionals $e(\mathbf{x}) = \sum_{n=p}^{\infty} G_n[\mathbf{x}]$, so $G_p[\mathbf{x}]$ is part of the error. As a consequence of the linearity of the expectation, Eq. (16) implies

$$\sum\nolimits_{n=p}^{\infty} E\left[\varphi_k(\mathbf{x})G_n[\mathbf{x}]\right] = 0 \quad \text{and} \quad \sum\nolimits_{n=p+1}^{\infty} E\left[\varphi_k(\mathbf{x})G_n[\mathbf{x}]\right] = 0 \quad (17)$$

for any $\phi_k$ of order less than $p$. The difference of both equations yields $E\left[\varphi_k(\mathbf{x})G_p[\mathbf{x}]\right] = 0$, so that $G_p[\mathbf{x}]$ must be orthogonal to any of the lower order basis functions, namely to all monomials with order smaller than $p$. □

## 3 Experiments

**Toy examples.** In our first experiment, we check whether our intuitions about higher-order statistics described in the introduction are captured by the proposed method. In particular, we expect that arbitrarily oriented lines can only be predicted using third-order statistics. As a consequence, we should need at least a second-order Wiener functional to predict lines correctly.

Our first test image (size $80 \times 110$, upper row in Fig. 1) contains only lines of varying orientations. Choosing a $5 \times 5$ neighbourhood, we predicted the central pixel using (11).

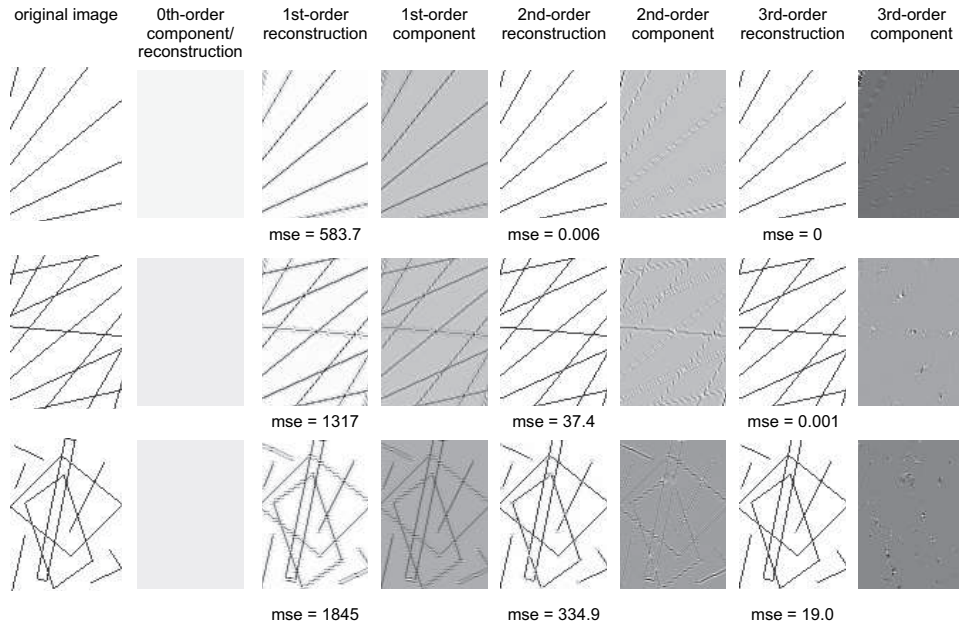

| original image | 0th-order component/ reconstruction | 1st-order reconstruction | 1st-order component | 2nd-order reconstruction | 2nd-order component | 3rd-order reconstruction | 3rd-order component |

Figure 1: Higher-order components of toy images. The image components of different orders are created by the corresponding Wiener functionals. They are added up to obtain the different orders of reconstruction. Note that the constant 0-order component and reconstruction are identical. The reconstruction error (mse) is given as the mean squared error between the true grey values of the image and the reconstruction. Although the linear first-order model seems to reconstruct the lines, this is actually not true since the linear model just smoothes over the image (note its large reconstruction error). A correct prediction is only obtained by adding a second-order component to the model. The third-order component is only significant at crossings, corners and line endings.

Models of orders $0 \ldots 3$ were learned from the image by extracting the maximal training set of $76 \times 106$ patches of size $5 \times 5$[7]. The corresponding image components of order 0 to 3 were computed according to (14). Note the different components generated by the Wiener functionals can also be negative. In Fig. 1, they are scaled to the gray values $[0..255]$. The behaviour of the models conforms to our intuition: the linear model cannot capture the line structure of the image thus leading to a large reconstruction error which drops to nearly zero when a second-order model is used. The additional small correction achieved by the third-order model is mainly due to discretization effects.

Similar to lines, we expect that we need at least a third-order model to predict crossings or corners correctly. This is confirmed by the second and third test image shown in the corresponding row in Fig. 1. Note that the third-order component is only significant at crossings, corners and line endings. The fourth- and fifth-order terms (not shown) have only negligible contributions. The fact that the reconstruction error does not drop to zero for the third image is caused by the line endings which cannot be predicted to a higher accuracy than one pixel.

**Application to natural images.** Are there further predictable structures in natural images that are not due to lines, crossings or corners? This can be investigated by applying our method to a set of natural images (an example of size $80 \times 110$ is depicted in Fig. 2). Our

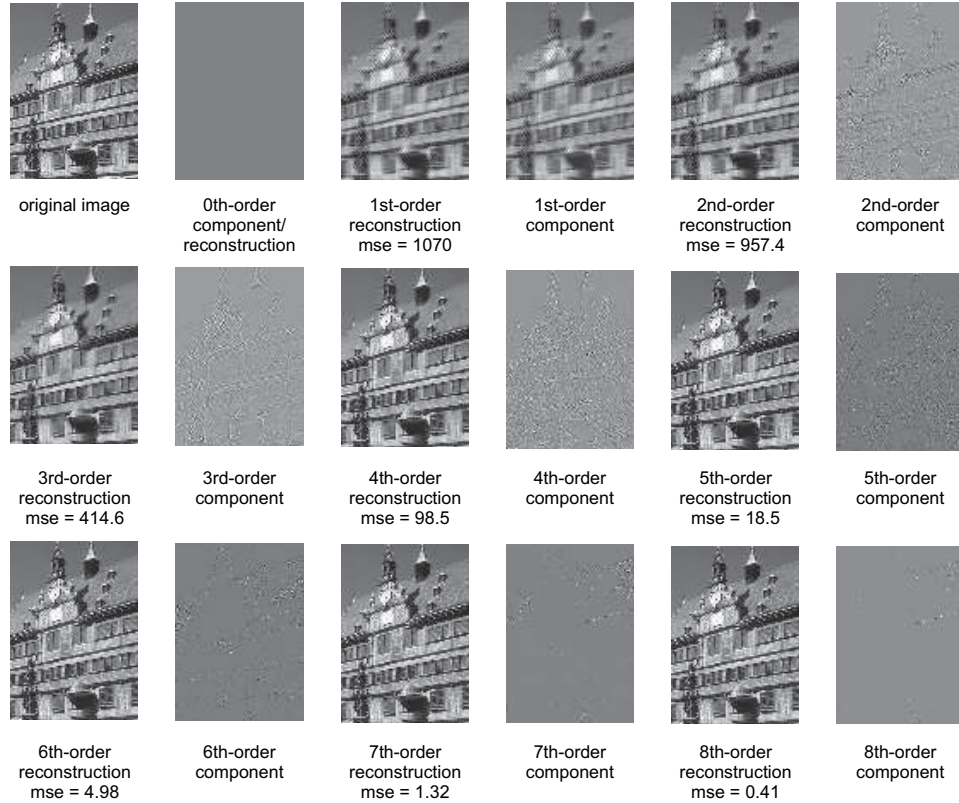

Figure 2: Higher-order components and reconstructions of a photograph. Interactions up to the fi fth order play an important role. Note that significant components become sparser with increasing model order.

results on a set of 10 natural images of size $50 \times 70$ show an an approximately exponential decay of the reconstruction error when more and more higher-order terms are added to the reconstruction (Fig. 3). Interestingly, terms up to order 5 still play a significant role, although the image regions with a significant component become sparser with increasing model order (see Fig. 2). Note that the nonlinear terms reduce the reconstruction error to almost 0. This suggests a high degree of higher-order redundancy in natural images that cannot be exploited by the usual linear prediction models.

## 4  Conclusion

The implicit estimation of Wiener functionals via polynomial kernels opens up new possibilities for the estimation of higher-order image statistics. Compared to the classical methods such as higher-order spectra, moments or cumulants, our approach avoids the combinatorial explosion caused by the exponential increase of the number of terms to be estimated and interpreted. When put into a predictive framework, multiplicative pixel interactions of different orders are easily visualized and conform to the intuitive notions about image structures such as edges, lines, crossings or corners.

There is no one-to-one mapping between the classical higher-order statistics and multiplicative pixel interactions. Any nonlinear Wiener functional, for instance, creates infinitely many correlations or cumulants of higher order, and often also of lower order. On the other

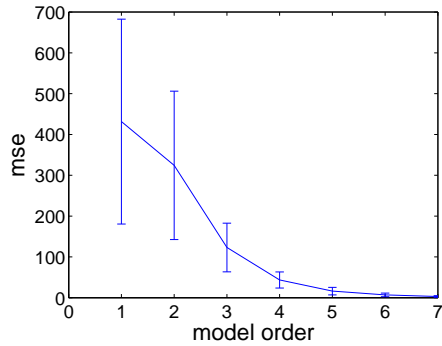

Figure 3: Mean square reconstruction error of models of different order for a set of 10 natural images.

hand, a Wiener functional of order $n$ produces only harmonic phase interactions up to order $n+1$, but sometimes also of lower orders. Thus, when one analyzes a classical statistic of a given order, one often cannot determine by which order of pixel interaction it was created. In contrast, our method is able to isolate image components that are created by a single order of interaction.

Although of preliminary nature, our results on natural images suggest an important role of statistics up to the fifth order. Most of the currently used low-level feature detectors such as edge or corner detectors maximally use third-order interactions. The investigation of fourth- or higher-order features is a field that might lead to new insights into the nature and role of higher-order image structures.

As often observed in the literature (e.g. [2][7]), our results seem to confirm that a large proportion of the redundancy in natural images is contained in the higher-order pixel interactions. Before any further conclusions can be drawn, however, our study needs to be extended in several directions: 1. A representative image database has to be analyzed. The images must be carefully calibrated since nonlinear statistics can be highly calibration-sensitive. In addition, the contribution of image noise has to be investigated. 2. Currently, only images up to 9000 pixels can be analyzed due to the matrix inversion required by Eq. 11. To accomodate for larger images, our method has to be reformulated in an iterative algorithm. 3. So far, we only considered $5 \times 5$-patches. To systematically investigate patch size effects, the analysis has to be conducted in a multi-scale framework.

## Footnotes

[1]Strictly speaking, the term *Wiener functional* is reserved for orthogonal Volterra functionals with respect to Gaussian input. Here, the term will be used for orthogonalized Volterra functionals with arbitrary input distributions.

[2]A similar approach has been taken by [1] using the inhomogeneous polynomial kernel $k_{inh}^{(p)}(\mathbf{x}_1, \mathbf{x}_2) = (1 + \mathbf{x}_1^\top \mathbf{x}_2)^p$. This kernel implies a map $\phi_{inh}$ into the same space of monomials, but it weights the degrees of the monomials differently as can be seen by applying the binomial theorem.

[3]for conditions on uniqueness of the solution, see [6]

[4]Note that this is different from the regularized approach used by [1]. If $\Omega$ is not zero, the resulting Volterra series are different from the Wiener series since they are not orthogonal with respect to the input.

[5]If $K$ is not invertible, $K^{-1}$ denotes the pseudo-inverse of $K$.

[6]assuming symmetrized Volterra kernels which can be obtained from any Volterra expanson.

[7]In contrast to the usual setting in machine learning, training and test set are identical in our case since we are not interested in *generalization* to other images, but in *analyzing* the higher-order components of the image at hand.

## References

[1] T. J. Dodd and R. F. Harrison. A new solution to Volterra series estimation. In *CD-Rom Proc. 2002 IFAC World Congress*, 2002.

[2] D. J. Field. What is the goal of sensory coding? *Neural Computation*, 6:559 – 601, 1994.

[3] M. O. Franz and B. Schölkopf. Implicit Wiener series. Technical Report 114, Max-Planck-Institut für biologische Kybernetik, Tübingen, June 2003.

[4] C. L. Nikias and A. P. Petropulu. *Higher-order spectra analysis*. Prentice Hall, Englewood Cliffs, NJ, 1993.

[5] M. Schetzen. *The Volterra and Wiener theories of nonlinear systems*. Krieger, Malabar, 1989.

[6] B. Schölkopf and A. J. Smola. *Learning with kernels*. MIT Press, Cambridge, MA, 2002.

[7] O. Schwartz and E. P. Simoncelli. Natural signal statistics and sensory gain control. *Nature Neurosc.*, 4(8):819 – 825, 2001.

[8] M. G. A. Thomson. Higher-order structure in natural scenes. *J. Opt.Soc. Am. A*, 16(7):1549 – 1553, 1999.

[9] M. G. A. Thomson. Beats, kurtosis and visual coding. *Network: Compt. Neural Syst.*, 12:271 – 287, 2001.
